# Using Feedforward Neural Networks to Monitor Alertness from Changes in EEG Correlation and Coherence

**Scott Makeig**
Naval Health Research Center, P.O. Box 85122
San Diego, CA   92186-5122

**Tzyy-Ping Jung**
Naval Health Research Center and
Computational Neurobiology Lab
The Salk Institute, P.O. Box 85800
San Diego, CA   92186-5800

**Terrence J. Sejnowski**
Howard Hughes Medical Institute and
Computational Neurobiology Lab
The Salk Institute, P.O. Box 85800
San Diego, CA   92186-5800

## Abstract

We report here that changes in the normalized electroencephalographic (EEG) cross-spectrum can be used in conjunction with feedforward neural networks to monitor changes in alertness of operators continuously and in near-real time. Previously, we have shown that EEG spectral amplitudes covary with changes in alertness as indexed by changes in behavioral error rate on an auditory detection task [6, 4]. Here, we report for the first time that increases in the frequency of detection errors in this task are also accompanied by patterns of increased and decreased spectral coherence in several frequency bands and EEG channel pairs. Relationships between EEG coherence and performance vary between subjects, but within subjects, their topographic and spectral profiles appear stable from session to session. Changes in alertness also covary with changes in correlations among EEG waveforms recorded at different scalp sites, and neural networks can also estimate alertness from correlation changes in spontaneous and unobtrusively-recorded EEG signals.

## 1   Introduction

When humans become drowsy, EEG scalp recordings of potential oscillations change dramatically in frequency, amplitude, and topographic distribution [3]. These changes are complex and differ between subjects [10]. Recently, we have shown

that using principal components analysis in conjunction with feedforward neural networks, minute-scale changes in performance on a sustained auditory detection task can be estimated in near real-time from changes in the EEG spectrum at one or more scalp channels [4, 6]. Here, we report, first, that loss of alertness during auditory detection task performance is also accompanied by changes in spectral coherence of EEG signals recorded at different scalp sites. The extent, topography, and frequency content of coherence changes linked to changes in alertness differ between subjects, but within subjects they appear stable from session to session. Second, since most coherence changes linked to alertness are not associated with significant phase differences, moving correlation measures applied to wideband or bandlimited EEG waveforms also covary with changes in alertness. Incorporating coherence and/or correlation information into neural network algorithms for estimating alertness from the EEG spectrum should enhance their accuracy and robustness and contribute to the design of practical neural human-system interfaces performing real-time monitoring of changes in operator alertness.

## 2    Methods

Concurrent EEG and behavioral data were collected for the purpose of developing a method of objectively monitoring the alertness of operators of complex systems [6]. Ten adult volunteers participated in three or more half-hour sessions during which they pushed one button whenever they detected an above-threshold auditory target stimulus (a brief increase in the level of the continuously-present background noise). To maximize the chance of observing alertness decrements, sessions were conducted in a small, warm, and dimly-lit experimental chamber, and subjects were instructed to keep their eyes closed.

Targets were 350 ms increases in the intensity of a 62 dB white noise background, 6 dB above their threshold of detectability, presented at random time intervals at a mean rate of 10/min. Short, and task-irrelevant probe tones of two frequencies (568 and 1098 Hz) were interspersed between the target noise bursts at 2-4 s intervals. EEG was collected from thirteen electrodes located at sites of the Internation 10-20 System, referred to the right mastoid, at a sampling rate of 312.5 Hz. A bipolar diagonal electrooculogram (EOG) channel was also recorded for use in eye movement artifact correction and rejection. Two sessions each from three of the subjects were chosen for analysis on the basis of their including more than 50 detection lapses.

A continuous performance measure, local error rate, was computed by convolving an irregularly-spaced performance index (hit=0/lapse=1) with a 95 s smoothing window advanced through the performance data in 1.64 s steps. Target hits were defined as targets responded to within a 100-3000 ms window; other targets were called lapses. After eye movement artifacts were removed from the data using a selective regression procedure [5], and data containing other large artifacts were rejected from analysis, complex EEG spectra were computed by advancing a 512-point (1.64 s) data window through the data in 0.41 s steps, multiplying by a Hanning window, and converting to frequency domain using an FFT.

Complex coherence was then computed for each channel pair in 1.64 s spectral epochs. In the coherence studies, error rate was smoothed with a bell-shaped Papoulis window; a 36 s rectangular window was used to smooth the coherence estimates. Finally, complex coherence was converted to coherence amplitude and phase and results were correlated with local error rate. A moving correlation measure between (1-20 Hz) bandlimited EEG waveforms was computed for each channel pair in a moving 1.64 s smoothing window, and then smoothed using a causal 95-s exponential window. The same window was used to smooth the error rate time series for the correlation studies.

## 3 Results

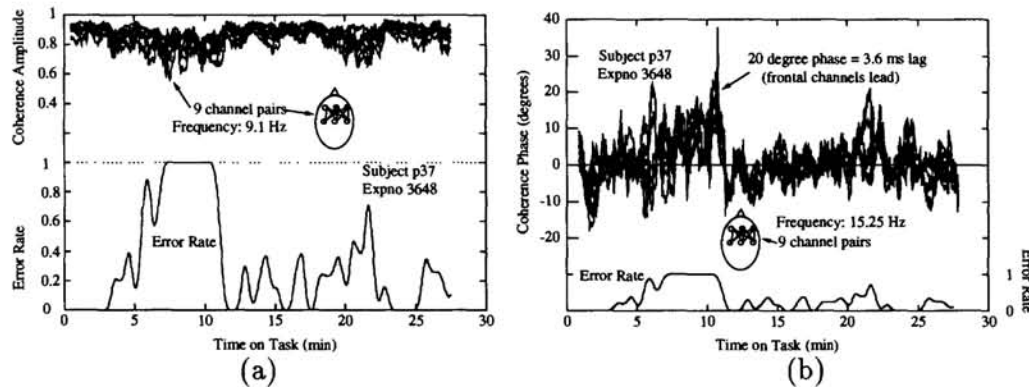

Figure 1: (a) Changes in coherence amplitude at 9.1 Hz (*upper traces*) are correlated with simultaneous changes in error rate during a half auditory detection task (*lower trace*) in nine indicated central-frontal channel pairs. (b) Concurrent changes in coherence phase at 15.25 Hz (*upper traces*) and local error rate (*lower trace*) for the same session and channel-pairs.

### 3.1  Relation of Coherence Changes to Detection Performance.

During the first 2-3 minutes of the session shown in Fig. 1a, the subject detected all targets presented, and coherence amplitudes remained high (0.9). In minutes 8-10, however, when the subject failed to make a single detection response(*lower trace*), coherence amplitude fell to as low as 0.6. Overall correlations for this session between the coherence and error rate time series in these channel pairs ranged from -0.590 to -0.776.

In the same session, coherence phase at 15 Hz also covaried with performance (Fig. 1b). During low-error portions of the session, there was no detectable coherence phase lag at 15 Hz within the same nine channel pairs, whereas while the subject performed poorly, a 20 degree phase lag appeared during which 15 Hz activity at frontal sites lead activity at frontal sites by 3 ms. Overall correlations for this session between coherence phase and error rate for these channel pairs ranged from 0.416 to 0.689. Correlations between coherence amplitude and error rate at 80 EEG frequencies (Fig. 2a, *upper traces*) included two broad bands of strong negative correlations (3-12 Hz and 15-20 Hz), while appreciable correlations between coherence phase and performance were confined to much narrower frequency bands (*lower traces*).

To estimate the significance of these coherence correlations, surrogate moving coherence records were collected 10 times using randomly-selected, asynchronous blocks of contiguous EEG data for each channel. Correlations between the resulting surrogate moving coherence time series and error rate were computed, and the 99.936th percentile of the distribution of (absolute) correlations was determined. For the subject whose data is shown here, this value was 0.485. Under conservative assumptions of complete independence of adjacent frequencies, this should give the (p=0.05) significance level for the maximum absolute correlation in each 80-bin correlation spectrum. (The heuristic estimate of this significance level from the surrogate data was 0.435). In the two sessions from this subject, however, more than 20% of all the 78 channel-pair coherence correlations were larger in absolute value than 0.485, implying that coherence amplitude changes at many scalp sites and frequencies are significantly related to changes in alertness in this subject.

## 3.2    Spectral and Topographic Stability

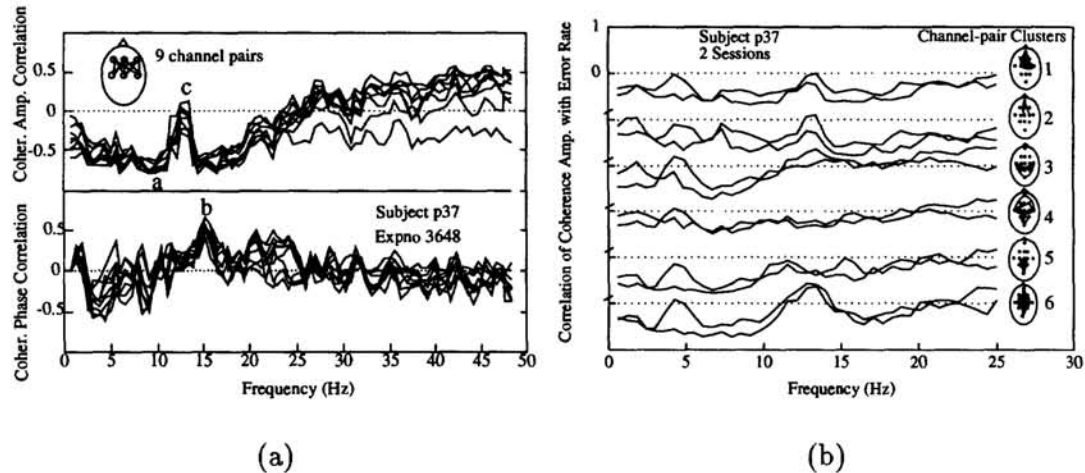

(a)                                                                    (b)

Figure 2: (a) Correlation spectra showing correlations between moving-average co-
herence and error rate for the same session and channel-pairs. Small letters 'a,b,c'
indicate the frequencies analyzed in Figs. 1 and 3. (b) Cluster analysis of correla-
tions between coherence amplitude and error rate at 41 frequencies (0.6 Hz to 25
Hz). Means of six sets of channel pairs derived from cluster analysis of 78 similar
coherence correlation spectra from all pairs of 13 scalp channels; superimposed on
the same means for a second session from the same subject.

The sign, size, and spectral and topographic structure of correlations between co-
herence amplitude and error rate at each frequency were stable across two sessions
for most channel pairs and frequency bands. Fig. 2b shows mean spectral correla-
tions in both sessions from the same subject for six clusters of similar channel-pair
correlation spectra identified by cluster analysis on results of the first session. Ex-
cept near 5 Hz, the size and structure of the correlation spectra for the second
session replicate results of the first session. The spectral stability of monotonic
relationships between EEG coherence and auditory detection performance suggests
that coherence may be used to predict changes in performance level from sponta-
neous EEG data collected continuously and unobtrusively from two or more scalp
channels.

## 3.3    EEG Waveform Correlations and Performance

In most cases, coherence phase lags in these data are small, and correlations be-
tween changes in phase lag and performance were insignificant. We therefore in-
vestigated whether moving-average correlations between band-limited EEG signals
in different scalp channels might also be used to predict changes in alertness, pos-
sibly at a lower computational cost, by studying the relationship between error
rate and changes in moving-average correlations of time-domain EEG waveforms
(1-20 Hz bandpass) in the same 6 sessions. Again, we found that the strength
and topographic structure of significant relationships between moving-correlation
and performance measures are stable within, and variable between subjects. For
each subject, we selected 8 EEG channel pairs whose moving-correlation time series
correlated most highly with error rate, and used these to train a multilinear regres-
sion network and three feedforward three-layer perceptrons to estimate error rate
from moving-average correlations. The feedforward neural networks had 3, 4, and
5 hidden units, respectively. Weights and biases of the network were adjusted using
the error backpropagation algorithm [9]. Conjugate gradient descent was used to
minimize the mean-squared error between network output and the actual error rate

time series. Cross-validation [7] was used to prevent the network from overfitting the training data. For each of the 6 training-testing session pairs and each neural network architecture, the time course of error rate was estimated five times using different random initial weights between -0.3 and 0.3. We tested the generalization ability of the models on second sessions from the same subjects. The procedure simulated potential real-world alertness monitoring applications in which pilot data for each operator would be used to train a network to estimate his or her alertness in subsequent sessions from unobtrusively-recorded EEG data.

Accuracy of error rate estimation in the test sessions was almost identical for neural networks with 3, 4, and 5 hidden units. Each was more accurate than multivariate linear regression. Figure 3 shows the time courses of actual and estimated error rate in one pair of training (*top panel*) and test sessions. Results for two other subjects were equivalent. Table 1 shows the average correlations and root-mean-squared estimation error between actual and estimated error rate time series for 6 sessions, 2 each on 3 subjects using a feedforward neural network with 3 hidden units. Results using 4 or 5 hidden units are equivalent. Diagonal cells show results for training sessions, off-diagonal cells for test sessions. The nonlinear adaptability of three-layer perceptrons give improved estimation performance over multivariate linear regression, reducing the RMS estimation error in the test sessions from 0.255 to 0.225 ($F(1,5) = 1234.29; p \leq 0.0001$), and increasing the mean correlation between actual and estimated error rate time series from 0.63 to 0.67 ($F(1,5) = 549.5; p \leq 0.0001$).

## 4   Discussion

Spectral coherence of EEG waveforms at different scalp sites has been measured for nearly 30 years [11]. and is the subject of a steadily increasing number of clinical, behavioral, and developmental EEG studies. Coherence values are known to be higher in sleep than in waking [8], and wake-sleep transitions have been noted to be preceded by increased coherence at some frequencies [2]. Our results, from data on three subjects performing a sustained auditory detection task under soporific conditions, suggest that during drowsiness, coherence may either increase or decrease, depending on the subject, analysis frequency, and electrode sites analyzed. However, in individual subjects the spectral and topographic structure of alertness-related coherence changes appears stable from session to session.

EEG correlation and coherence are intimately related: changes in moving-average correlations of EEG waveforms reflect changes in broad-band, zero-lag coherence of activity at the same sites. The possibility of using moving-average correlation measures of electrophysiological activity to monitor state changes in animals was discussed by Arduini [1], but to our knowledge this approach has not previously been applied to human EEG.

The origin and function of nonstationarity in EEG synchrony are not yet understood. Decreased EEG coherence during drowsiness might result from inactivation of subcortical brain systems coordinating activity in separate cortical EEG generators during wakefulness, or from emergence of drowsiness-related EEG activity projecting preferentially to one part of the scalp surface. Similarly, increases in coherence in drowsiness might either result from increased synchrony between cortical generators, or from volume conduction of enhanced activity generated at a single cortical or subcortical site. Measuring changes in EEG coherence and correlation during other cognitive tasks give clues to the possible role of variable EEG synchrony in brain and cognitive dynamics.

We are now investigating to what extent moving EEG coherence and/or correlation

measures, in combination with spectral amplitude measures [4], will allow practical, robust, continuous, and near-real time estimation of alertness level in auditory detection and other task environments.

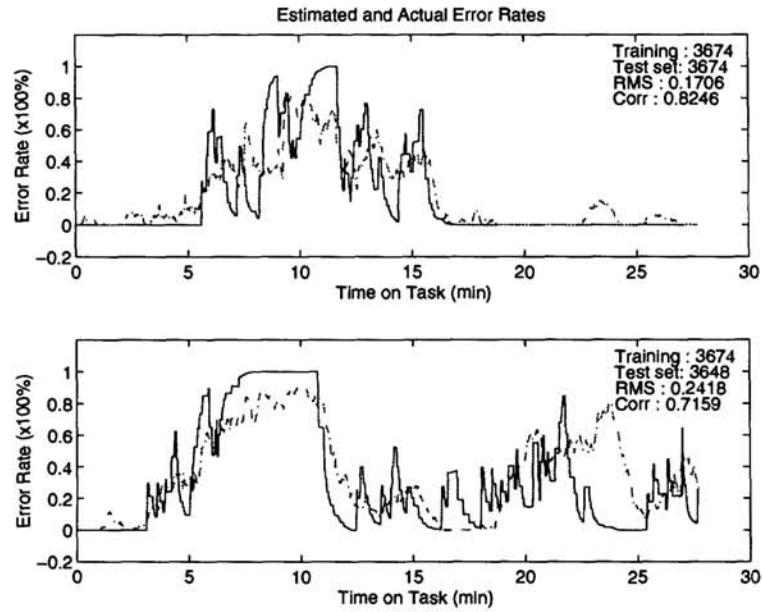

Figure 3: Changes in detection rate (95-s exponential window) and their estimate using a feedforward three-layer perceptron on moving correlations between (1-20 Hz) band passed EEG signals for 8 selected pairs of 7 scalp channels. The top panel shows the training session, the bottom panel the testing session. Solid lines show the actual error rate time course; dashed lines, the estimate. Correlation and RMS error between the two are indicated.

Table 1: The results of alertness monitoring using moving EEG pairwise correlation.

Subject A

| Test | Training Set | |
|------|------|------|
| set | 3648 | 3674 |
| 3648 | rms : 0.17 | rms : 0.26 |
|      | corr : 0.87 | corr : 0.68 |
| 3674 | rms : 0.21 | rms : 0.17 |
|      | corr : 0.73 | corr: 0.83 |

Subject B

| Test | Training Set | |
|------|------|------|
| set | 3654 | 3656 |
| 3654 | rms : 0.17 | rms : 0.22 |
|      | corr : 0.83 | corr : 0.73 |
| 3656 | rms : 0.25 | rms : 0.14 |
|      | corr : 0.54 | corr: 0.76 |

Subject C

| Test | Training Set | |
|------|------|------|
| set | 3665 | 3673 |
| 3665 | rms : 0.19 | rms : 0.23 |
|      | corr : 0.76 | corr : 0.65 |
| 3673 | rms : 0.18 | rms : 0.17 |
|      | corr : 0.67 | corr: 0.70 |

## Acknowledgments

This work was supported by a grant (ONR.Reimb.30020.6429) to the Naval Health Research Center by the Office of Naval Research. The views expressed in this article are those of the authors and do not reflect the official policy or position of the Department of the Navy, Department of Defense, or the U.S. Government. We acknowledge the contributions of Keith Jolley, F.Scot Elliott, and Mark Postal in collecting and processing the data, and thank Tony Bell for suggestions.

## References

[1] Arduini A.. 1979. In-phase brain activity and sleep. *Electroencephalog clin Neurophysiol* 47, 441-9

[2] Borodkin SM, Grindel' OM, Boldyreva GN, Zaitsev VA & Luk'ianov V.I. 1987. Dynamics of the spectral-coherent characteristics of the human EEG in healthy subjects and brain pathology. *Zh Vyssh Nerv Deiat* 37, 22-30

[3] Davis H., Davis P.A., Loomis A.L., Harvey E.N., & Hobart G. 1938. Human brain potentials during the onset of sleep. *J Neurophysiol* 1, 24-38

[4] Jung T-P, Makeig S., Stensmo M., & Sejnowski T. Estimating alertness from the EEG power spectrum, submitted for publication.

[5] Kenemans J.L., Molenaar P.C.M., Verbaten M.N. & Slangen J.L. 1991. Removal of the ocular artifact from the EEG: a comparison of time and frequency domain methods with simulated and real data. *Psychophysiolog* 28, 114-121

[6] Makeig S. & Inlow M. 1993. Lapses in alertness: Coherence of fluctuations in performance and EEG spectrum. *Electroencephalog clin Neurophysiol* 86, 23-35

[7] Morgan N., & Bourlard H. 1990. Generalization and parameter estimation in feedforward nets: some experiments. *Neural Information Processing Systems,* 2, 630-637.

[8] Nielsen T, Abel A, Lorrain D, & Montplaisir J. 1990. Interhemispheric EEG coherence during sleep and wakefulness in left- and right-handed subjects. *Brain and Cognition* 14, 113-25

[9] Rumelhart D, Hinton G, & Williams R, 1986. Learning internal representation by error propagation, *Parallel distributed processing, Chap. 8.*

[10] Santamaria J. & Chiappa K.H. 1987. The EEG of drowsiness in normal adults. *J Clin Neurophysiol* 4, 327-82

[11] Walter D.O. 1968. Coherence as a measure of relationship between EEG records. *Electroencephalog clin Neurophysiol* 24, 282